# The cerebellum chip:
# an analog VLSI implementation of a
# cerebellar model of classical conditioning

**Constanze Hofstötter, Manuel Gil, Kynan Eng,**
**Giacomo Indiveri, Matti Mintz, Jörg Kramer[*] and Paul F. M. J. Verschure**

Institute of Neuroinformatics
University/ETH Zurich
CH-8057 Zurich, Switzerland
*pfmjv@ini.phys.ethz.ch*

## Abstract

We present a biophysically constrained cerebellar model of classical conditioning, implemented using a neuromorphic analog VLSI (aVLSI) chip. Like its biological counterpart, our cerebellar model is able to control adaptive behavior by predicting the precise timing of events. Here we describe the functionality of the chip and present its learning performance, as evaluated in simulated conditioning experiments at the circuit level and in behavioral experiments using a mobile robot. We show that this aVLSI model supports the acquisition and extinction of adaptively timed conditioned responses under real-world conditions with ultra-low power consumption.

## 1 Introduction

The association of two correlated stimuli, an initially neutral conditioned stimulus (CS) which predicts a meaningful unconditioned stimulus (US), leading to the acquisition of an adaptive conditioned response (CR), is one of the most essential forms of learning. Pavlov introduced the classical conditioning paradigm in the early 20th century to study associative learning (Pavlov 1927). In classical conditioning training an animal is repeatedly exposed to a CS followed by a US after a certain inter-stimulus interval (ISI). The animal learns to elicit a CR matched to the ISI, reflecting its knowledge about an association between the CS, US, and their temporal relationship. Our earlier software implementation of a

---

[*]Jörg Kramer designed the cerebellum chip that was first tested at the 2002 Telluride Neuromorphic Engineering Workshop. Tragically, he died soon afterwards while hiking on Telescope Peak on 24 July, 2002.

biophysically constrained model of the cerebellar circuit underlying classical conditioning (Verschure and Mintz 2001; Hofstötter et al. 2002) provided an explanation of this phenomenon by assuming a negative feedback loop between the cerebellar cortex, deep nucleus and inferior olive. It could acquire and extinguish correctly timed CRs over a range of ISIs in simulated classical conditioning experiments, as well as in associative obstacle avoidance tasks using a mobile robot. In this paper we present the analog VLSI (aVLSI) implementation of this cerebellum model – the cerebellum chip – and the results of chip-level and behavioral robot experiments.

## 2   The model circuit and aVLSI implementation

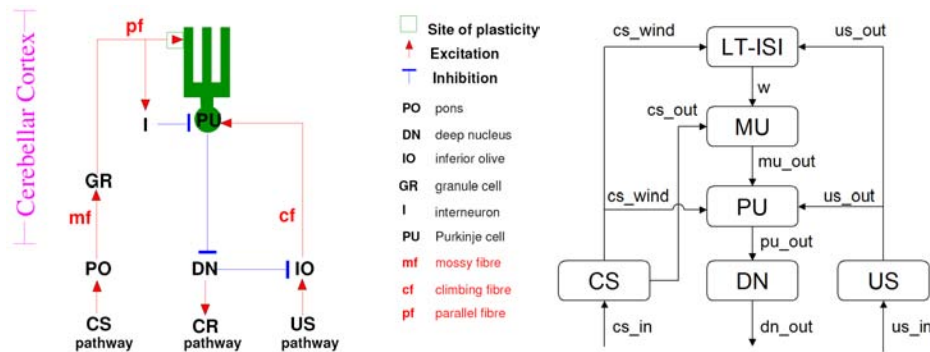

Figure 1: Anatomy of the cerebellar model circuit (left) and the block diagram of the corresponding chip (right).

The model (Figure 1) is based on the identified cerebellar pathways of CS, US and CR (Kim and Thompson 1997) and includes four key hypotheses which were implemented in the earlier software model (Hofstötter et al. 2002):

1.   CS related parallel fiber (pf) and US related climbing fiber (cf) signals converge at Purkinje cells (PU) in the cerebellum (Steinmetz et al. 1989). The direction of the synaptic changes at the pf-PU-synapse depends on the temporal coincidence of pf and cf activity. Long-term depression (LTD) is induced by pf activity followed by cf activity within a certain time interval, while pf activity alone induces long-term potentiation (LTP) (Hansel et al. 2001).

2.   A prolonged second messenger response to pf stimulation in the dendrites of PU constitutes an eligibility trace from the CS pathway (Sutton and Barto 1990) that bridges the ISI (Fiala et al. 1996).

3.   A microcircuit (Ito 1984) comprising PU, deep nucleus (DN) and inferior olive (IO) forms a negative feedback loop. Shunting inhibition of IO by DN blocks the reinforcement pathway (Thompson et al. 1998), thus controlling the induction of LTD and LTP at the pf-PU-synapse.

4.   DN activity triggers behavioral CRs (McCormick and Thompson 1984). The inhibitory PU controls DN activity by a mechanism called rebound excitation (Hesslow 1994): When DN cells are disinhibited from PU input, their

membrane potential slowly repolarises and spikes are emitted if a certain threshold is reached. Thereby, the correct timing of CRs results from the adaptation of a pause in PU spiking following the CS.

In summary, in the model the expression of a CR is triggered by DN rebound excitation upon release from PU inhibition. The precise timing of a CR is dependent on the duration of an acquired pause in PU spiking following a CS. The PU response is regulated by LTD and LTP at the pf-PU-synapse under the control of a negative feedback loop comprising DN, PU and IO.

We implemented an analog VLSI version of the cerebellar model using a standard $1.6\mu m$ CMOS technology, and occupying an area of approximately $0.25$ mm$^2$. A block diagram of the hardware model is shown in Figure 1. The *CS* block receives the conditioned stimulus and generates two signals: an analog long-lasting, slowly decaying trace (*cs_out*) and an equally long binary pulse (*cs_wind*). Similarly, the *US* block receives an unconditioned stimulus and generates a fast pulse (*us_out*). The two pulses *cs_wind* and *us_out* are sent to the *LT-ISI* block that is responsible for perfoming LTP and LTD, upregulating or downregulating the synaptic weight signal *w*. This signal determines the gain by which the *cs_out* trace is multiplied in the *MU* block. The output of the multiplier *MU* is sent on to the *PU* block, together with the *us_out* signal. It is a linear integrate-and-fire neuron (the axon-hillock circuit) connected to a constant current source that produces regular spontaneous activity. The current source is gated by the digital *cf_wind* signal, such that the spontaneous activity is shut off for the duration of the *cs_out* trace.

The chip allowed one of three learning rules to be connected. Experiments showed that an ISI-dependent learning rule with short ISIs resulting in the strongest LTD was the most useful (Kramer and Hofstötter 2002). Two elements were added to adapt the model circuit for real-world robot experiments. Firstly, to prevent the expression of a CR after a US had already been triggered, an inhibitory connection from IO to CRpathway was added. Secondly, the transduction delay (TD) from the aVLSI circuit to any effectors (e.g. motor controls of a robot) had to be taken into account, which was done by adding a delay from DN to IO of 500ms.

The chip's power consumption is conservatively estimated at around 100 $\mu$W (excluding off-chip interfacing), based on measurements from similar integrate-and-fire neuron circuits (Indiveri 2003). This figure is an order of magnitude lower than what could be achieved using conventional microcontrollers (typically 1-10 mW), and could be improved further by optimising the circuit design.

## 3   Simulated conditioning experiments

The aim of the "in vitro" simulated conditioning experiments was to understand the learning performance of the chip. To obtain a meaningful evaluation of the performance of the learning system for both the simulated conditioning experiments and the robot experiments, the measure of *effective CRs* was used. In acquisition experiments CS-US pairs are presented with a fixed ISI. Whenever a CR occurs that precedes the US, the US signal is not propagated to PU due to the inhibitory connection from DN to IO. Thus in the context of acquisition experiments a CR is defined as effective if it prevents the occurrence of a US spike

at PU. In contrast, in robot experiments an effective CR is defined at the behavioral level, including only CRs that prevent the US from occurring.

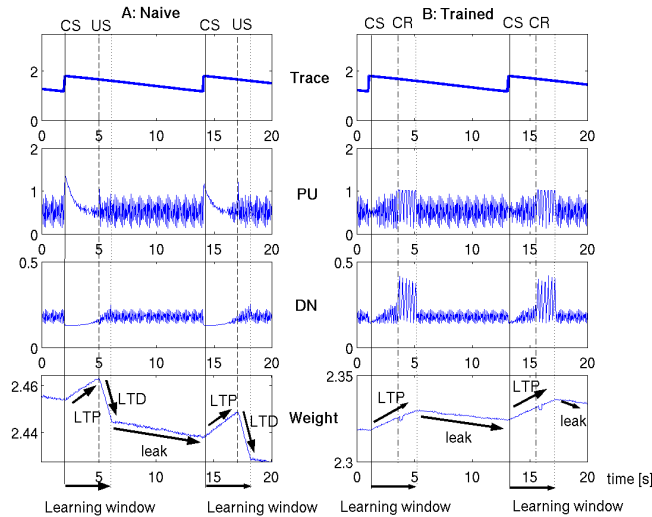

Figure 2: Learning related response changes in the cerebellar aVLSI chip. The most relevant neural responses to a CS-US pair (ISI of 3s, ITI of 12s) are presented for a trial before (naive) significant learning occurred and when a correctly timed CR is expressed (trained). US-related pf and CS/CR-related cf signals are indicated by vertical lines passing through the subplots. A CS-related pf-signal evokes a prolonged response in the pf-PU-synapse, the CS-trace (*Trace* subplot). While an active CS-trace is present, an inhibitory element (I) is active which inactivates an element representing the spontaneous activity of PU (Hofstötter et al. 2002). (A) The US-related cf input occurs while there is an active CS-trace (*Trace* subplot), in this case following the CS with an ISI of 3s. LTD predominates over LTP under these conditions (*Weight* subplot). Because the PU membrane potential (PU) remains above spiking threshold, PU is active and supplies constant inhibition to DN (DN) while in the CS-mode. Thus, DN cannot repolarize and remains inactive so that no CR is triggered. (B) Later in the experiment, the synaptic weight of the pf-PU-synapse (Weight) has been reduced due to previous LTD. As a result, following a CS-related pf input, the PU potential (*PU* subplot) falls below the spiking threshold, which leads to a pause in PU spiking. The DN membrane potential repolarises, so that rebound spikes are emitted (*DN* subplot). This rebound excitation triggers a CR. DN inhibition of IO prevents US related cf-activity. Thus, although a US signal is still presented to the circuit, the reinforcing US pathway is blocked. These conditions induce only LTP, raising the synaptic weight of the pf-PU-synapse (*Weight* subplot).

The results we obtained were broadly consistent with those reported in the biological literature (Ito 1984; Kim and Thompson 1997). The correct operation of the circuit can be seen in the cell traces illustrating the properties of the aVLSI circuit components before significant learning (Figure 2 A), and after a CR is expressed (Figure 2B). Long-term acquisition experiments (25 blocks of 10 trials

each over 50 minutes) showed that chip functions remained stable over a long time period. In each trial the CS was followed by a US with a fixed ISI of 3s; the inter trial interval (ITI) was 12s. The number of effective CRs shows an initial fast learning phase followed by a stable phase with higher percentages of effective CRs (Figure 3B). In the stable phase the percentage of effective CRs per block fluctuates around 80-90%. There are fluctuations of up to 500ms in the CR latency caused by the interaction of LTD and LTP in the stable phase, but the average CR latency remains fairly constant.

Figure 4 shows the average of five acquisition experiments (5 blocks of 10 trials per experiment) for ISIs of 2.5s, 3s and 3.5s. The curves are similar in shape to the ones in the long-term experiment. The CR latency quickly adjusts to match the ISI and remains stable thereafter (Figure 4A). The effect of the ISI-dependent learning rule can be seen in two ways: firstly, the shorter the ISI, the faster the stable phase is reached, denoting faster learning. Secondly, the shorter the ISI, the better the performance in terms of percentage of effective CRs (Figure 4B). The parameters of the chip were tuned to optimally encode short ISIs in the range of 1.75s to 4.5s. Separate experiments showed that the chip could also adapt rapidly to changes in the ISI within this range after initial learning.

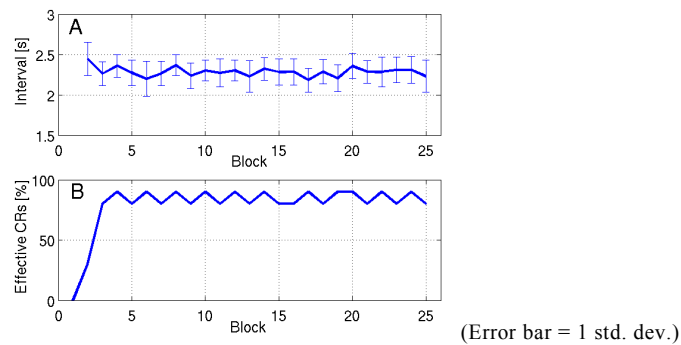

(Error bar = 1 std. dev.)

Figure 3: Long-term changes in CR latency (A) and % effective CRs (B) per block of 10 CSs during acquisition. Experiment length = 50min., ISI = 3s, ITI = 12s.

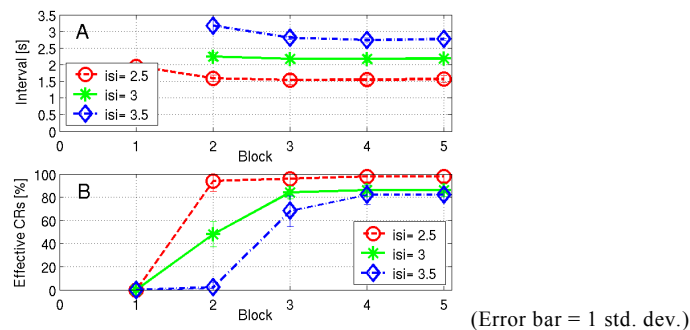

(Error bar = 1 std. dev.)

Figure 4: Average of five acquisition experiments per block of 10 CSs for ISIs of 2.5s (○), 3s (*) and 3.5s (◊). (A) Avg. CR latency. (B) Avg. % effective CRs.

## 4  Robot associative learning experiments

The "in vivo" learning capability of the chip was evaluated by interfacing it to a robot and observing its behavior in an unsupervised obstacle avoidance task. Experiments were performed using a Khepera microrobot (K-team, Lausanne, Switzerland, Figure 5A) in a circular arena with striped walls (Figure 5C). The robot was equipped with 6 proximal infra-red (IR) sensors (Figure 5B). Activation of these sensors (US) due to a collision triggered a turn of ~110° in the opposite direction (UR). A line camera (64 pixels x 256 gray-levels) constituted the distal sensor, with detection of a certain spatial frequency (~0.14 periods/degree) signalling the CS. Visual CSs and collision USs were conveyed to CSpathway and USpathway on the chip. The activation of CRpathway triggered a motor CR: a 1s long regression followed by a turn of ~180°. Communication between the chip and the robot was performed using Matlab on a PC. The control program could be downloaded to the robot's processor, allowing the robot to act fully autonomously. In each experiment, the robot was placed in the circular arena exploring its environment with a constant speed of ~4 cm/s. A spatial frequency CS was detected at some distance when the robot approached the wall, followed by a collision with the wall, stimulating the IR sensors and thus triggering a US. Consequently the CS was correlated with the US, predicting it. The ISIs of these stimuli were variable, due to noise in sensor sampling, and variations in the angle at which the robot approached the wall.

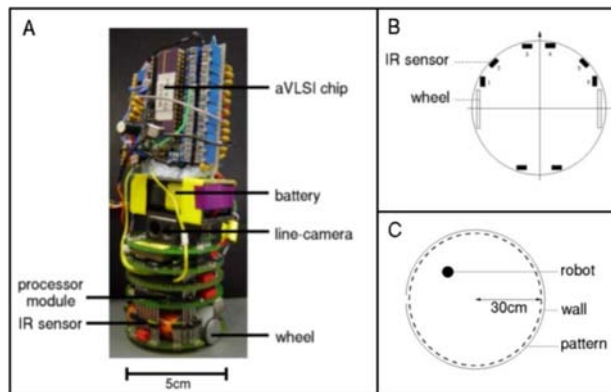

Figure 5: (A) Khepera microrobot with aVLSI chip mounted on top. (B) Only the forward sensors were used during the experiments. (C) The environment: a 60cm diameter circular arena surrounded by a 15cm high wall. A pattern of vertical, equally sized black and white bars was placed on the wall.

Associative learning mediated by the cerebellum chip significantly altered the robot's behavior in the obstacle avoidance task (Figure 6) over the course of each experiment. In the initial learning phase, the behavior was UR driven: the robot drove forwards until it collided with the wall, only then performing a turn (Figure 6A1). In the trained phase, the robot usually turned just before it collided with the wall (Figure 6A2), reducing the number of collisions. The positions of the robot when a CS, US or CR event occurred in these two phases are shown in Figure 6B1

and B2. The CRs were not expressed immediately after the CSs, but rather with a CR latency adjusted to just prevent collisions (USs). Not all USs were avoided in the trained phase due to some excessively short ISIs (Figure 7) and normal extinction processes over many unreinforced trials. After the learning phase the percentage of effective CRs fluctuated between 70% and 100% (Figure 7).

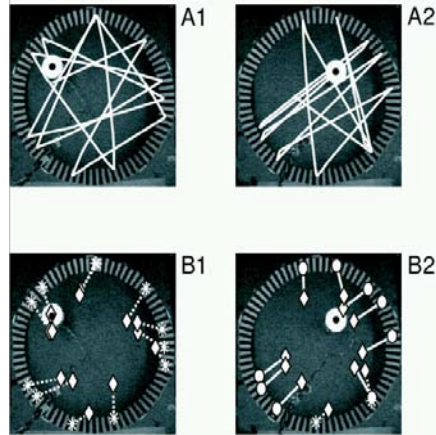

Figure 6: Learning performance of the robot. (Top row) Trajectories of the robot. The white circle with the black dot in the center indicates the beginning of trajectories. (Bottom row) The same periods of the experiment examined at the circuit level: ◊ = CS, * = US, ○ = CR. (A1, B1) Beginning of the experiment (CS 3-15). (A2, B2) Later in the experiment (CS 32-44).

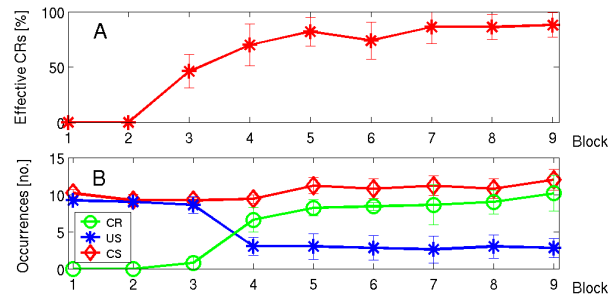

Figure 7: Trends in learning behavior (average of 5 experiments, 25 min. each). 90 CSs were presented in each experiment. Error bars indicate one standard deviation. (A) Average percentage of effective CRs over 9 blocks of 10 CSs. (B) Number of CS occurrences (◊), US occurrences (*) and CR occurrences (○).

## 5   Discussion

We have presented one of the first examples of a biologically constrained model of learning implemented in hardware. Our aVLSI cerebellum chip supports the acquisition and extinction of adaptively timed responses under noisy, real world

conditions. These results provide further evidence for the role of the cerebellar circuit embedded in a synaptic feedback loop in the learning of adaptive behavior, and pave the way for the creation of artefacts with embedded ultra low-power learning capabilities.

# 6   References

Fiala, J. C., Grossberg, S. and Bullock, D. (1996). *Metabotropic glutamate receptor activation in cerebellar Purkinje cells as substrate for adaptive timing of the classical conditioned eye-blink response.* Journal of Neuroscience **16**: 3760-3774.

Hansel, C., Linden, D. J. and D'Angelo, E. (2001). *Beyond parallel fiber LTD, the diversity of synaptic and nonsynaptic plasticity in the cerebellum.* Nature Neuroscience **4**: 467-475.

Hesslow, G. (1994). *Inhibition of classical conditioned eyeblink response by stimulation of the cerebellar cortex in decerebrate cat.* Journal of Physiology **476**: 245-256.

Hofstötter, C., Mintz, M. and Verschure, P. F. M. J. (2002). *The cerebellum in action: a simulation and robotics study.* European Journal of Neuroscience **16**: 1361-1376.

Indiveri, G. (2003). *A low-power adaptive integrate-and-fire neuron circuit.* IEEE International Symposium on Circuits and Systems, Bangkok, Thailand, **4:** 820-823.

Ito, M. (1984). *The modifiable neuronal network of the cerebellum.* Japanese Journal of Physiology **5**: 781-792.

Kim, J. J. and Thompson, R. F. (1997). *Cerebellar circuits and synaptic mechanisms involved in classical eyeblink conditioning.* Trends in the Neurosciences **20**(4): 177-181.

Kim, J. J. and Thompson, R. F. (1997). *Cerebellar circuits and synaptic mechanisms involved in classical eyeblink conditioning.* Trend. Neurosci. **20**: 177-181.

Kramer, J. and Hofstötter, C. (2002). An aVLSI model of cerebellar mediated associative learning. Telluride Workshop, CO, USA.

McCormick, D. A. and Thompson, R. F. (1984). *Neuronal response of the rabbit cerebellum during acquisition and performance of a classical conditioned nictitating membrane-eyelid response.* J. Neurosci. **4**: 2811-2822.

Pavlov, I. P. (1927). *Conditioned Reflexes*, Oxford University Press.

Steinmetz, J. E., Lavond, D. G. and Thompson, R. F. (1989). *Classical conditioning in rabbits using pontine nucleus stimulation as a conditioned stimulus and inferior olive stimulation as an unconditioned stimulus.* Synapse **3**: 225-233.

Sutton, R. S. and Barto, A. G. (1990). Time derivate models of Pavlovian Reinforcement Learning and Computational Neuroscience: Foundations of Adaptive Networks., MIT press**:** chapter 12, 497-537.

Thompson, R. F., Thompson, J. K., Kim, J. J. and Shinkman, P. G. (1998). *The nature of reinforcement in cerebellar learning.* Neurobiology of Learning and Memory **70**: 150-176.

Verschure, P. F. M. J. and Mintz, M. (2001). *A real-time model of the cerebellar circuitry underlying classical conditioning: A combined simulation and robotics study.* Neurocomputing **38-40**: 1019-1024.
